# An EM Algorithm for Localizing Multiple Sound Sources in Reverberant Environments

**Michael I. Mandel, Daniel P. W. Ellis**
LabROSA, Dept. of Electrical Engineering
Columbia University
New York, NY
{mim,dpwe}@ee.columbia.edu

**Tony Jebara**
Dept. of Computer Science
Columbia University
New York, NY
jebara@cs.columbia.edu

## Abstract

We present a method for localizing and separating sound sources in stereo recordings that is robust to reverberation and does not make any assumptions about the source statistics. The method consists of a probabilistic model of binaural multi-source recordings and an expectation maximization algorithm for finding the maximum likelihood parameters of that model. These parameters include distributions over delays and assignments of time-frequency regions to sources. We evaluate this method against two comparable algorithms on simulations of simultaneous speech from two or three sources. Our method outperforms the others in anechoic conditions and performs as well as the better of the two in the presence of reverberation.

## 1 Introduction

Determining the direction from which a sound originated using only two microphones is a difficult problem. It is exacerbated by the presence of sounds from other sources and by realistic reverberations, as would be found in a classroom. A related and equally difficult problem is determining in which regions of a spectrogram a sound is observable, the so-called time-frequency mask, useful for source separation [1]. While humans can solve these problems well enough to carry on conversations in the canonical "cocktail party", current computational solutions are less robust. Either they assume sound sources are statistically stationary, they assume anechoic conditions, or they require an array with at least as many microphones as there are sources to be localized.

The method proposed in this paper takes a probabilistic approach to localization, using the psychoacoustic cue of interaural phase difference (IPD). Unlike previous approaches, this EM algorithm estimates true probability distributions over both the direction from which sounds originate and the regions of the time-frequency plane associated with each sound source. The basic assumptions that make this possible are that a single source dominates each time-frequency point and that a single delay and amplification cause the difference in the ears' signals at a particular point.

By modelling the observed IPD in this way, this method overcomes many of the limitations of other systems. It is able to localize more sources than it has observations, even in reverberant environments. It makes no assumptions about the statistics of the source signal, making it well suited to localizing speech, a highly non-Gaussian and non-stationary signal. Its probabilistic nature also facilitates the incorporation of other probabilistic cues for source separation such as those obtained from single-microphone computational auditory scene analysis.

Many comparable methods are also based on IPD, but they first convert it into interaural time difference. Because of the inherent $2\pi$ ambiguity in phase differences, this mapping is one-to-one only up to a certain frequency. Our system, however, is able to use observations across the entire frequency

range, because even though the same phase difference can correspond to multiple delays, a particular delay corresponds unambiguously to a specific phase difference at every frequency.

We evaluate our system on the localization and separation of two and three simultaneous speakers in simulated anechoic and reverberant environments. The speech comes from the TIMIT acoustic-phonetic continuous speech corpus [2], the anechoic simulations use the head related transfer functions described in [3], and the reverberant simulations use the binaural classroom impulse responses described in [4]. We use four metrics to evaluate our system, the root mean square localization error, the mutual information between the estimated mask and a ground truth mask, the signal to noise ratio of separated speech from [5], and the W-disjoint orthogonality metric of [1]. Our EM approach outperformed Yilmaz and Rickard's DUET algorithm [1] and Aarabi's PHAT-histogram [6] in anechoic situations, and performed comparably to PHAT-histogram in reverberation.

## 1.1 Previous work

Many systems exist for localizing sounds using a microphone array, e.g. [7]. These systems can be quite accurate, but this accuracy requires physical bulk, special hardware to synchronize many recordings, and tedious calibration procedures. They isolate signals in reverberation using directional filtering, which becomes more selective only through the addition of further microphones.

Because of the structure and abilities of the human auditory system, researchers have paid particular attention to the two-microphone case. Roman et al. [5] make empirical models of the timing and level differences for combinations of two sources in known positions synthesized with anechoic head-related transfer functions (HRTFs). They then classify each time-frequency cell in their auditory-based representation, creating a binary time-frequency mask which contains the cells that appear to be dominated by the target source.

Yilmaz and Rickard [1] studied the interaction of speech signals in the time-frequency plane, concluding that multiple speech signals generally do not overlap much in both time and frequency. They also conclude, as in [5], that the best ground truth mask includes only points in which the signal to noise ratio is 0dB or greater. They propose a method for localization that maps IPD to delay before aggregating information and thus cannot use information at higher frequencies. It is designed for anechoic and noise-free situations and subsequently its accuracy suffers in more realistic settings.

Aarabi [6] and Rennie [8] focus on localizing sounds. Aarabi's method, while quite simple, is still one of the most accurate methods for localizing many simultaneous sound sources, even in reverberation. Rennie refined this approach with an EM algorithm for performing the same process probabilistically. A limitation of both algorithms, however, is the assumption that a single source dominates each analysis window, as compared to time-frequency masking algorithms which allow different sources to dominate different frequencies in the same analysis window.

## 2 Framework

For the purposes of deriving this model we will examine the situation where one sound source arrives at two spatially distinct microphones or ears. This will generalize to the assumption that only a single source arrives at each time-frequency point in a spectrogram, but that different points can contain different sources.

Denote the sound source as $s(t)$, and the signals received at the left and right ears as $\ell(t)$ and $r(t)$, respectively. The two received signals will have some delay and some gain relative to the source, in addition to a disruption due to noise. For this model, we assume a convolutive noise process, because it fits our empirical observations, it is easy to analyze, and in general is it is very similar to the additive noise processes that other authors assume. The various signals are then related by,

$$\ell(t) = a_\ell s(t - \tau_\ell) * n_\ell(t) \qquad r(t) = a_r s(t - \tau_r) * n_r(t). \qquad (1)$$

The ratio of the short-time Fourier transforms, $\mathcal{F}\{\cdot\}$, of both equations is the interaural spectrogram,

$$X_{IS}(\omega, t) \equiv \frac{L(\omega, t)}{R(\omega, t)} = \alpha(\omega, t)e^{\phi(\omega, t)} = e^{a - j\omega\tau} N(\omega, t), \qquad (2)$$

where $\tau = \tau_\ell - \tau_r$, $N(\omega, t) = \frac{N_\ell(\omega, t)}{N_r(\omega, t)} = \frac{\mathcal{F}\{n_\ell(t)\}}{\mathcal{F}\{n_r(t)\}}$, and $a = \log \frac{a_\ell}{a_r}$. This equivalence assumes that $\tau$ is much smaller than the length of the window over which the Fourier transform is taken, a

condition easily met for dummy head recordings with moderately sized Fourier transform windows. For example, in our experiments the maximum delay was 0.75ms, and the window length was 64ms.

As observed in [9], $N(\omega, t)$, the noise in the interaural spectrogram of a single source, is uni-modal and approximately identically distributed for all frequencies and times. Using the standard rectangular-to-polar change of coordinates, the noise can be separated into independent magnitude and phase components. The magnitude noise is approximately log-normal, while the phase noise has a circular distribution with tails heavier than the von Mises distribution. In this work, we ignore the magnitude noise and approximate the phase noise with a mixture of Gaussians, all with the same mean. This approximation includes the distribution's heavy tailed characteristic, but ignores the circularity, meaning that the variance of the noise is generally underestimated. A true circular distribution is avoided because its maximum likelihood parameters cannot be found in closed form.

## 3  Derivation of EM algorithm

The only observed variable in our model is $\phi(\omega, t)$, the phase difference between the left and right channels at frequency $\omega$ and time $t$. While $2\pi$ ambiguities complicate the calculation of this quantity, we use $\phi(\omega, t) = \arg\left(\frac{L(\omega, t)}{R(\omega, t)}\right)$ so that it stays within $(-\pi, \pi]$. For similar reasons, we define $\hat{\phi}(\omega, t; \tau) = \arg\left(\frac{L(\omega, t)}{R(\omega, t)} e^{j\omega\tau}\right)$ as a function of the observation.

Our model of the interaural phase difference is a mixture over sources, delays, and Gaussians. In particular, we have $I$ sources, indexed by $i$, each of which has a distribution over delays, $\tau$. For this model, the delays are discretized to a grid and probabilities over them are computed as a multinomial. This discretization gives the most flexible possible distribution over $\tau$ for each source, but since we expect sources to be compact in $\tau$, a unimodal parametric distribution could work. Experiments approximating Laplacian and Gaussian distributions over $\tau$, however, did not perform as well at localizing sources or creating masks as the more flexible multinomial.

For a particular source, then, the probability of an observed delay is

$$p(\phi(\omega, t) \,|\, i, \tau) = p_{\angle N}(\hat{\phi}(\omega, t; \tau)) \tag{3}$$

where $p_{\angle N}(\cdot)$ is the probability density function of the phase noise, $N(\omega, t)$, described above. We approximate this distribution as a mixture of $J$ Gaussians, indexed by $j$ and centered at 0,

$$p(\phi(\omega, t) \,|\, i, \tau) = \sum_{j=1}^{J} p(j \,|\, i) \mathcal{N}(\hat{\phi}(\omega, t; \tau) \,|\, 0, \sigma_{ij}^2) \tag{4}$$

In order to allow parameter estimation, we define hidden indicator variables $z_{ij\tau}^{\omega t}$ such that $z_{ij\tau}^{\omega t} = 1$ if $\phi(\omega, t)$ comes from Gaussian $j$ in source $i$ at delay $\tau$, and 0 otherwise. There is one indicator for each observation, so $\sum_{ij\tau} z_{ij\tau}^{\omega t} = 1$ and $z_{ij\tau}^{\omega t} \geq 0$.

The estimated parameters of our model are thus $\psi_{ij\tau} \equiv p(i, j, \tau)$, a third order tensor of discrete probabilities, and $\sigma_{ij}$, the variances of the various Gaussians. For convenience, we define $\theta \equiv \{\psi_{ij\tau}, \sigma_{ij} \quad \forall i, j, \tau\}$.

Thus, the total log-likelihood of our data, including marginalization over the hidden variables, is:

$$\log p(\phi(\omega, t) \,|\, \theta) = \sum_{\omega t} \log \sum_{ij\tau} \psi_{ij\tau} \mathcal{N}(\hat{\phi}(\omega, t; \tau) \,|\, 0, \sigma_{ij}^2). \tag{5}$$

This log likelihood allows us to derive the E and M steps of our algorithm. For the E step, we compute the expected value of $z_{ij\tau}^{\omega t}$ given the data and our current parameter estimates,

$$\nu_{ij\tau}(\omega, t) \equiv E\{z_{ij\tau}^{\omega t} \,|\, \phi(\omega, t), \theta\} = p(z_{ij\tau}^{\omega t} = 1 \,|\, \phi(\omega, t), \theta) = \frac{p(z_{ij\tau}^{\omega t} = 1, \phi(\omega, t) \,|\, \theta)}{p(\phi(\omega, t) \,|\, \theta)} \tag{6}$$

$$= \frac{\psi_{ij\tau} \mathcal{N}(\hat{\phi}(\omega, t; \tau) \,|\, 0, \sigma_{ij}^2)}{\sum_{ij\tau} \psi_{ij\tau} \mathcal{N}(\hat{\phi}(\omega, t; \tau) \,|\, 0, \sigma_{ij}^2)} \tag{7}$$

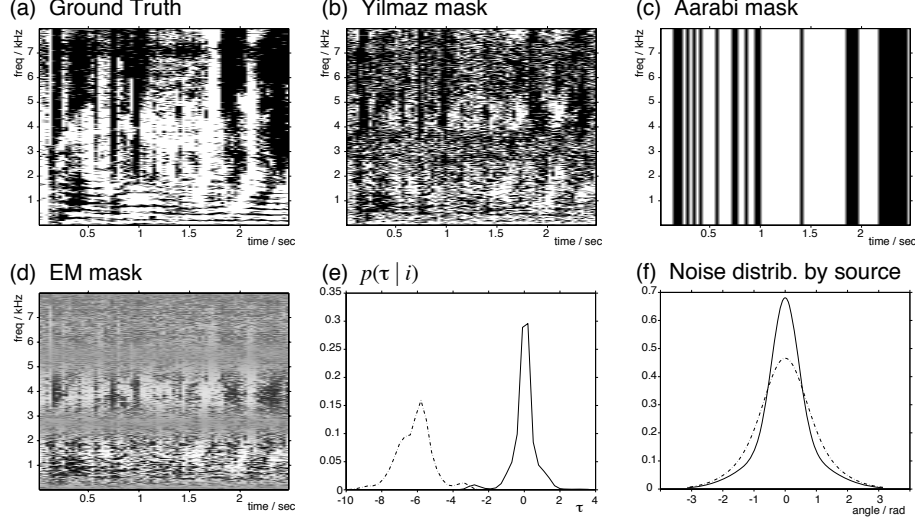

Figure 1: Example parameters estimated for two speakers located at $0°$ and $45°$ in a reverberant classroom. (a) Ground truth mask for speaker 1, (b) mask estimated by Yilmaz's algorithm, (c) mask estimated by Aarabi's algorithm, (d) mask estimated by EM algorithm, (e) probability distribution over $\tau$ for each speaker $p(\tau \mid i)$ estimated by EM algorithm, (f) probability distribution over phase error for each speaker, $p(\phi(\omega, t) \mid i, \tau)$ estimated by EM algorithm.

For the M step, we first compute the auxiliary function $Q(\theta \mid \theta_s)$, where $\theta$ is the set of parameters over which we wish to maximize the likelihood, and $\theta_s$ is the estimate of the parameters after $s$ iterations of the algorithm.

$$Q(\theta \mid \theta_s) = c + \sum_{\omega t} \sum_{ij\tau} \nu_{ij\tau}(\omega, t) \log p(\phi(\omega, t), z_{ij\tau}^{\omega t} \mid \theta) \qquad (8)$$

where $c$ does not depend on $\theta$. Since $Q$ is concave in $\theta$, we can maximize it by taking derivatives with respect to $\theta$ and setting them equal to zero while also including a Lagrange multiplier to enforce the constraint that $\sum_{ij\tau} \psi_{ij\tau} = 1$. This results in the update rules

$$\psi_{ij\tau} = \frac{1}{\Omega T} \sum_{\omega t} \nu_{ij\tau}(\omega, t) \qquad \sigma_{ij}^2 = \frac{\sum_{\omega t} \sum_{\tau} \nu_{ij\tau}(\omega, t) \hat{\phi}(\omega, t; \tau)^2}{\sum_{\omega t} \sum_{\tau} \nu_{ij\tau}(\omega, t)}. \qquad (9)$$

Note that we are less interested in the joint distribution $\psi_{ij\tau} = p(i, j, \tau)$ than other distributions derived from it. Specifically, we are interested in the marginal probability of a point's coming from source $i$, $p(i)$, the distributions over delays and Gaussians conditioned on the source, $p(\tau \mid i)$ and $p(j \mid i)$, and the probability of each time-frequency point's coming from each source, $M_i(\omega, t)$. To calculate these masks, we marginalize $p(z_{ij\tau}^{\omega t} \mid \phi(\omega, t), \theta)$ over $\tau$ and $j$ to get

$$M_i(\omega, t) \equiv p(z_i^{\omega t} \mid \phi(\omega, t), \theta) = \sum_{j\tau} p(z_{ij\tau}^{\omega t} \mid \phi(\omega, t), \theta) = \sum_{j\tau} \nu_{ij\tau}(\omega, t). \qquad (10)$$

See Figure 1(d)-(f) for an example of the parameters estimated for two speakers located at $0°$ and $45°$ in a reverberant classroom.

## 4   Experiments

In order to evaluate our system, we simulated speech in anechoic and reverberant noise situations by convolving anechoic speech samples with binaural impulse responses. We used speech from the TIMIT acoustic-phonetic continuous speech corpus [2], a dataset of utterances spoken by 630 native American English speakers. Of the 6300 utterances in the database, we chose 15 at random to use in

our evaluation. To allow the speakers to be equally represented in each mixture, we normalized all of the signals by their average energies before convolving them with the binaural impulse responses.

The anechoic binaural impulse responses came from Algazi et al. [3], a large effort to record head-related transfer functions for many different individuals. Impulse response measurements were taken over the sphere surrounding subjects' heads at 25 different azimuths and 50 different elevations. The measurements we used were for the KEMAR dummy head with small ears, although the dataset contains impulse responses for around 50 individuals.

The reverberant binaural impulse responses we used were recorded by Shinn-Cunningham et al. in a real classroom [4]. These measurements were also made with a KEMAR dummy head, although a different actual unit was used. Measurements were taken at four different positions in the classroom, three distances from the subject, seven directions, and three repetitions of each measurement. We used the measurements taken in the middle of the classroom with the sources at a distance of 1 m from the subject.

Our method has a number of parameters that need to be set. Perhaps the most important part of running the algorithm is the initialization. We initialized it by setting $p(\tau \mid i)$ to discrete approximations to Gaussians centered at the $I$ largest peaks in the average cross-correlation. The other parameters are numerical. Following [1], we use a 1024 point window, which corresponds to 64 ms at 16 kHz. We chose $J$, the number of Gaussians in the noise GMM, to be 2, striking a balance between model flexibility and computational cost. Since the log likelihood increases monotonically with each EM iteration, we chose to stop after 10, when improvements in log likelihood generally became insignificant. Finally, we discretized $\tau$ to 31 values linearly spaced between -0.9375 ms and 0.9375 ms.

## 4.1 Comparison algorithms

We compare the performance of the time-frequency masks and the localization accuracy of our algorithm with those of two other algorithms. The first is Yilmaz and Rickard's DUET algorithm from [1], although it had to be modified slightly to accommodate our recordings. In order to estimate the interaural time and level differences of the signals in a mixture, DUET creates a two-dimensional histogram of them at every point in the interaural spectrogram. It then smooths the histogram and finds the $I$ largest peaks, which should correspond to the $I$ sources.

The interaural parameter calculation of DUET requires that the interaural phase of a measurement unambiguously translates to a delay. The maximum frequency at which this is possible is $\frac{c}{2d}$ where $c$ is the speed of sound and $d$ is the distance between the two microphones. The authors in [1] choose a fixed sampling rate and adjust the distance between their free-standing microphones to prevent ambiguity. In the case of our KEMAR recordings, however, the distance between the two ears is fixed at approximately 0.15 m and since the speed of sound is approximately 340 m/s, we must lower the maximum frequency from 8000 to 1150 Hz. Even though the frequencies used to estimate the interaural parameters are limited, a time-frequency mask can still be computed for all frequencies. See Figure 1(b) for an example of such a mask estimated by DUET.

We also implemented Aarabi's PHAT-histogram technique from [6], augmented to create time-frequency masks. The algorithm localizes multiple simultaneous sources by cross-correlating the left and right channels using the Phase Transform (PHAT) for each frame of the interaural spectrogram. This gives point estimates of the delay at each frame, which are pooled over all of the frames of the signal into a histogram. The $I$ largest peaks in this histogram are assumed to be the interaural delays of the $I$ sources. While not designed to create time-frequency masks, one can be constructed that simply assigns an entire frame to the source from which its delay originates. See Figure 1(c) for an example mask estimated by PHAT-histogram.

As discussed in the next section, we compare these algorithms using a number of metrics, some of which admit baseline masks. For power-based metrics, we include ground truth and random masks in the comparison as baselines. The ground truth, or 0 dB, mask is the collection of all time-frequency points in which a particular source is louder than the mixture of all other sources, it is included to measure the maximum improvement achievable by an algorithmically created mask. The random mask is created by assigning each time-frequency point to one of the $I$ sources at random, it is included to measure the performance of the simplest possible masking algorithm.

## 4.2 Performance measurement

Measuring performance of localization results is straightforward, we use the root-mean-square error. Measuring the performance of time-frequency masks is more complicated, but the problem has been well studied in other papers [5, 1, 10]. There are two extremes possible in these evaluations. The first of which is to place equal value on every time-frequency point, regardless of the power it contains, e.g. the mutual information metric. The second is to measure the performance in terms of the amount of energy allowed through the mask or blocked by it, e.g. the SNR and WDO metrics.

To measure performance valuing all points equally, we compute the mutual information between the ground truth mask and the predicted mask. Each mask point is treated as a binary random variable, so the mutual information can easily be calculated from their individual and joint entropies. In order to avoid including results with very little energy, the points in the lowest energy decile in each band are thrown out before calculating the mutual information. One potential drawback to using the mutual information as a performance metric is that it has no fixed maximum, it is bounded below by 0, but above by the entropy of the ground truth mask, which varies with each particular mixture. Fortunately, the entropy of the ground truth mask was close to 1 for almost all of the mixtures in this evaluation.

To measure the signal to noise ratio (SNR), we follow [5] and take the ratio of the amount of energy in the original signal that is passed through the mask to the amount of energy in the mixed signal minus the original signal that is passed through the mask. Since the experimental mixtures are simulated, we have access to the original signal. This metric penalizes masks that eliminate signal as well as masks that pass noise. A similar metric is described in [1], the W-disjoint orthogonality (WDO). This is the signal to noise ratio in the mixture the mask passes through, multiplied by a (possibly negative) penalty term for eliminating signal energy.

When evaluated on speech, energy based metrics tend to favor systems with better performance at frequencies below 500 Hz, where the energy is concentrated. Frequencies up to 3000 Hz, however, are still important for the intelligibility of speech. In order to more evenly distribute the energy across frequencies and thus include the higher frequencies more equally in the energy-based metrics, we apply a mild high pass pre-emphasis filter to all of the speech segments. The experimental results were quite similar without this filtering, but the pre-emphasis provides more informative scoring.

## 4.3 Results

We evaluated the performance of these algorithms in four different conditions, using two and three simultaneous speakers in reverberant and anechoic conditions. In the two source experiments, the target source was held at $0°$, while the distracter was moved from $5°$ to $90°$. In the three source experiments, the target source was held at $0°$ and distracters were located symmetrically on either side of the target from $5°$ to $90°$. The experiment was repeated five times for each separation, using different utterances each time to average over any interaction peculiarities. See Figure 2 for plots of the results of all of the experiments.

Our EM algorithm performs quite well at localization. Its root mean square error is particularly low for two-speaker and anechoic tests, and only slightly higher for three speakers in reverberation. It does not localize well when the sources are very close together, i.e. within $5°$, most likely because of problems with its automatic initialization. At such a separation, two cross-correlation peaks are difficult to discern. Performance also suffers slightly for larger separations, most likely a result of greater head shadowing. Head shadowing causes interaural intensity differences at high frequencies which change the distribution of IPDs, and violate our model's assumption that phase noise is identically distributed across frequencies.

It also performs well at time-frequency masking, more so for anechoic simulations than reverberant. See Figure 1(d) for an example time-frequency mask in reverberation. Notice that the major features follow the ground truth, but much detail is lost. Notice also the lower contrast bands in this figure at 0, 2.7, and 5.4 kHz corresponding to the frequencies at which the sources have the same IPD, modulo $2\pi$. For any particular relative delay between sources, there are frequencies which provide no information to distinguish one from the other. Our EM algorithm, however, can distinguish between the two because the soft assignment in $\tau$ uses information from many relative delays.

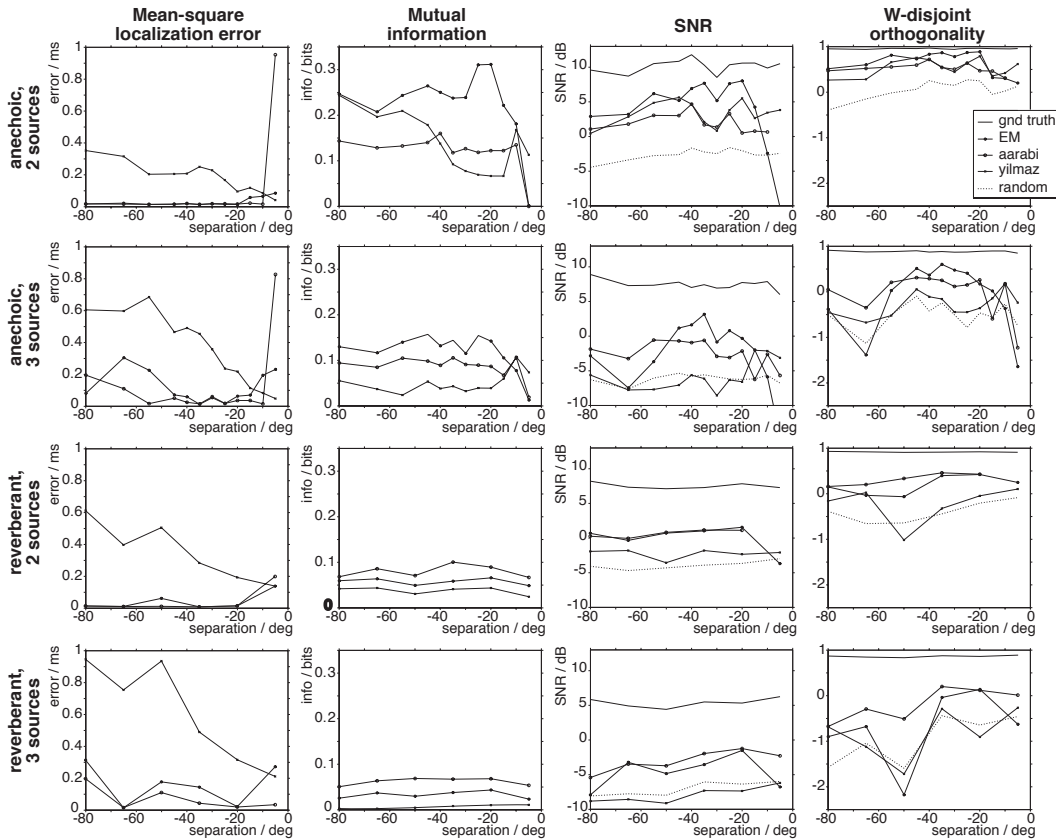

Figure 2: Experimental results for four conditions (rows) compared using four metrics (columns). First row: two sources, anechoic; second row: three sources, anechoic; third row: two sources, reverberant; fourth row: three sources, reverberant.

The EM approach always performs as well as the better of the other two algorithms and outperforms them both in many situations. Its localization performance is comparable to PHAT-histogram in two-speaker conditions and slightly worse in three-speaker conditions. DUET suffers even in anechoic, two-source situations, possibly because it was designed for free-standing microphones as opposed to dummy head recordings. Its performance decreases further as the tasks become more difficult.

The advantage of our method for masking, however, is particularly clear in anechoic conditions, where it has the highest mutual information at all angles and the highest SNR and WDO at lower angles. In reverberant conditions, the mutual information between estimated masks and ground truth masks becomes quite low, but PHAT-histogram comes out slightly ahead. Comparing SNR measurements in reverberation, PHAT-histogram and the EM approach perform similarly, with DUET trailing. In WDO, however, PHAT-histogram performs best, with EM and DUET performing similarly to the random mask.

## 5 Conclusions and Future Work

We have derived and demonstrated an expectation-maximization algorithm for probabilistic source separation and time-frequency masking. Using the interaural phase delay, it is able to localize more sources than microphones, even in the reverberation found in a typical classroom. It does not depend on any assumptions about sound source statistics, making it well suited for such non-stationary signals as speech and music. Because it is probabilistic, it is straightforward to augment the feature representation with other monaural or binaural cues.

There are many directions to take this project in the future. Perhaps the largest gain in signal separation accuracy could come from the combination of this method with other computational auditory scene analysis techniques [11, 12]. A system using both monaural and binaural cues should surpass the performance of either approach alone. Another binaural cue that would be easy to add is IID caused by head shadowing and pinna filtering, allowing localization in both azimuth and elevation.

This EM algorithm could also be expanded in a number of ways by itself. A minimum entropy prior [13] could be included to keep the distributions of the various sources separate from one another. In addition, a parametric, heavy tailed model could be used instead of the current discrete model to ensure unimodality of the distributions and enforce the separation of different sources. Along the same lines, a variational Bayes model could be used with a slightly different parameterization to treat all of the parameters probabilistically, as in [14]. Finally, we could relax the independence constraints between adjacent time-frequency points, making a Markov random field. Since sources tend to dominate regions of adjacent points in both time and frequency, the information at its neighbors could help a particular point localize itself.

## Acknowledgments

The authors would like to thank Barbara Shinn-Cunningham for sharing her lab's binaural room impulse response data with us and Richard Duda for making his lab's head-related transfer functions available on the web. This work is supported in part by the National Science Foundation (NSF) under Grants No. IIS-0238301, IIS-05-35168, CCR-0312690, and IIS-0347499. Any opinions, findings and conclusions or recommendations expressed in this material are those of the author(s) and do not necessarily reflect the views of the NSF.

## References

[1] Ozgur Yilmaz and Scott Rickard. Blind separation of speech mixtures via time-frequency masking. *IEEE Transactions on signal processing*, 52(7):1830–1847, July 2004.

[2] J. S. Garofolo, L. F. Lamel, W. M. Fisher, J. G. Fiscus, D. S. Pallett, and N. L. Dahlgren. DARPA TIMIT acoustic phonetic continuous speech corpus CDROM, 1993.

[3] V. R. Algazi, R. O. Duda, D. M. Thompson, and C. Avendano. The CIPIC HRTF database. In *Proc 2001 IEEE Workshop on Applications of Signal Processing to Audio and Electroacoustics*, pages 99–102, Oct 2001.

[4] Barbara Shinn-Cunningham, Norbert Kopco, and Tara J. Martin. Localizing nearby sound sources in a classroom: Binaural room impulse responses. *Journal of the Acoustical Society of America*, 117:3100–3115, 2005.

[5] Nicoleta Roman, DeLiang Wang, and Guy J. Brown. A classification-based cocktail party processor. In *Proceedings of Neural Information Processing Systems*, 2003.

[6] Parham Aarabi. Self-localizing dynamic microphone arrays. *IEEE transactions on systems, man, and cybernetics*, 32(4), November 2002.

[7] M. Brandstein and H. Silverman. A practical methodology for speech source localization with microphone arrays. *Computer, Speech, and Language*, 11(2):91–126, April 1997.

[8] Steven J. Rennie. Robust probabilistic TDOA estimation in reverberant environments. Technical Report PS1-TR-2005-011, University of Toronto, February 2005.

[9] Michael I. Mandel and Daniel P. W. Ellis. A probability model for interaural phase difference. *Workshop on Statistical and Perceptual Audio Processing (SAPA)*, 2006.

[10] Ron Weiss and Daniel P. W. Ellis. Estimating single-channel source separation masks: relevance vector machine classifiers vs pitch-based masking. *Workshop on Statistical and Perceptual Audio Processing (SAPA)*, 2006.

[11] Martin Cooke and Daniel P. W. Ellis. The auditory organization of speech and other sources in listeners and computational models. *Speech Communication*, 35(3–4):141–177, 2001.

[12] Sam Roweis. One microphone source separation. In *Proceedings of Neural Information Processing Systems 13*, pages 793–799, 2000.

[13] Matthew Brand. Pattern discovery via entropy minimization. In *Proceedings of Artificial Intelligence and Statistics*, 1999.

[14] Matthew J. Beal, Hagai Attias, and Nebojsa Jojic. Audio-video sensor fusion with probabilistic graphical models. In *ECCV (1)*, pages 736–752, 2002.
